# A comparison of Image Processing Techniques for Visual Speech Recognition Applications

**Michael S. Gray**
Computational Neurobiology Laboratory
The Salk Institute
San Diego, CA   92186-5800

**Terrence J. Sejnowski**
Computational Neurobiology Laboratory
The Salk Institute
San Diego, CA   92186-5800

**Javier R. Movellan**[*]
Department of Cognitive Science
Institute for Neural Computation
University of California San Diego

## Abstract

We examine eight different techniques for developing visual representations in machine vision tasks. In particular we compare different versions of principal component and independent component analysis in combination with stepwise regression methods for variable selection. We found that local methods, based on the statistics of image patches, consistently outperformed global methods based on the statistics of entire images. This result is consistent with previous work on emotion and facial expression recognition. In addition, the use of a stepwise regression technique for selecting variables and regions of interest substantially boosted performance.

## 1   Introduction

We study the performance of eight different methods for developing image representations based on the statistical properties of the images at hand. These methods are compared on their performance on a visual speech recognition task. While the representations developed are specific to visual speech recognition, the methods themselves are general purpose and applicable to other tasks. Our focus is on low-level data-driven methods based on the statistical properties of relatively untouched images, as opposed to approaches that work with contours or highly processed versions of the image. Padgett [8] and Bartlett [1] systematically studied statistical methods for developing representations on expression recognition tasks. They found that local wavelet-like representations consistently outperformed global representations, like eigenfaces. In this paper we also compare local versus global representations. The main differences between our work and that in [8] and [1]

---

[*] To whom correspondence should be addressed.

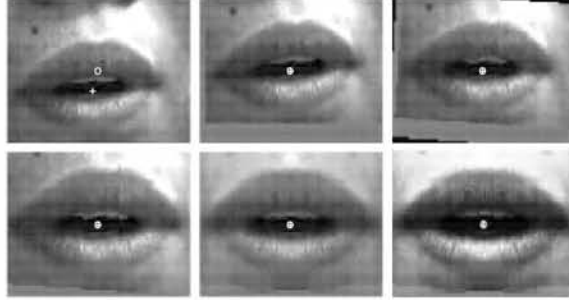

Figure 1: The normalization procedure. In each panel, the "+" indicates the center of the lips, and the "o" indicates the center of the image. The location of the lips was automatically determined using Luettin et al. point density model for lip tracking: (1) Original image; (2) The center of the lips was translated to the center of the image; (3) The image was rotated in the plane to horizontal; (4) The lips were scaled to a constant reference width; (5) The image was symmetrized relative to the vertical midline; (6) The intensity was normalized using a logistic gain control procedure.

are: (1) We use image sequences while they used static images; (2) Our work involves images of the mouth region while their work involves images of the entire face; (3) Our recognition engine is a bank of hidden Markov model while theirs is a backpropagation network [8] and a nearest neighbor classifier [1]. In addition to the comparison of local and global representations, we propose an unsupervised method for automatically selecting regions and variables of interest.

## 2 Preprocessing and Recognition Engine

The task was recognition of the words "one", "two", "three" and "four" from the Tulips1 [7] database. The database consists on movies of 12 subjects each uttering the digits in English twice. While the number of words is limited, the database is challenging due to differences in illumination conditions, ethnicity and gender of the subjects. Image preprocessing consisted of the following steps: First the contour of the outer lips were tracked using point distribution models, a data-driven technique based on analysis of the gray-level statistics around lip contours [5]. The lip images were then normalized for translation and rotation. This was accomplished by first padding the image on all sides with 25 rows or columns of zeros, and modulating the images in the spatial frequency domain. The images were symmetrized with respect to the vertical axis going through the center of the lips. This makes the final representation more robust to horizontal changes in illumination. The images were cropped to 65 pixels vertically × 87 pixels horizontally (see Figure 1) and their intensity was normalized using logistic gain control [7]. Eight different techniques were used on the normalized database each of which developed a different image basis. For each of these techniques the following steps were followed: (1) *Projection*: For each image in the database we compute the coordinates $x(t)$ of the image with respect to the image bases developed using each of the eight techniques; (2) *Temporal differentiation*: For each time step we compute the vectors $\delta(t) = x(t) - x(t-1)$, where $x(t)$ represents the coordinate vector of image presented at time $t$; (3) *Gain control*: Each component of $x(t)$ and $\delta(t)$ is independently scaled using a logistic gain control function matched to the mean and variance of each component across an entire movie [7]. This results in a form of soft histogram equalization; (4)

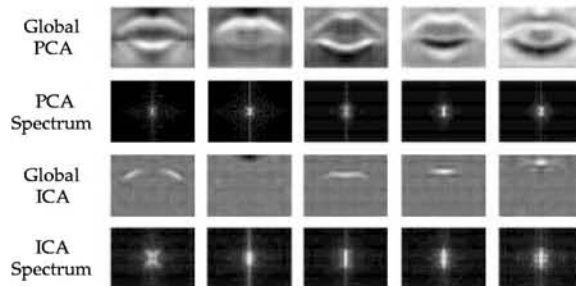

Figure 2: Global decompositions for the normalized image dataset. Row 1: Global kernels of principal component analysis ordered with first eigenimage on left. Row 2: Log magnitude spectrum of eigenimages. Row 3: Global pixel space independent component kernels ordered according to projected variance. Row 4: Log magnitude spectrum of global independent components.

*Recognition:* The scaled $x(t)$ and $\delta(t)$ coefficients are fed to the HMM recognition engine.

## 3 Global Methods

We first evaluated the performance of techniques based on the statistics of the entire lip images as opposed to portions of it. This global approach has been shown to provide good performance on face recognition [9], expression recognition [2], and gender recognition tasks [4]. In particular we compared the performance of principal component analysis (PCA) and two different versions of independent component analysis (ICA).

### 3.1 Global PCA:

We tried image bases that consisted of the first 50, 100 and 150 eigenvectors of the pixelwise covariance matrix. Best results were obtained with the first 50 principal components (which accounted for 94.6% of the variance) and are the only ones reported here. The top row of Figure 2 shows the first 5 eigenvectors displayed as images, their magnitude spectrum is shown in the second row. These eigenimages have most of their energy localized in low and horizontal spatial frequencies and are typically non-local in the spatial domain (i.e., have non-zero energy distributed over the whole image).

### 3.2 Global ICA:

The goal of Infomax ICA is to transform an input random vector such that the entropy of the output vector is maximized [3]. The main differences between ICA and PCA are: (1) ICA maximizes the joint entropy of the outputs, while PCA maximizes the sum of their variance; (2) PCA provides orthogonal basis vectors, while ICA basis vectors need not be orthogonal; (3) PCA outputs are always uncorrelated, but may not be statistically independent. ICA attempts to extract independent outputs, not just uncorrelated. We tried two different ICA approaches:

**ICA I:** This method results in a non-orthogonal transformation of the bases developed via PCA. While such transformations do not change the underlying space of

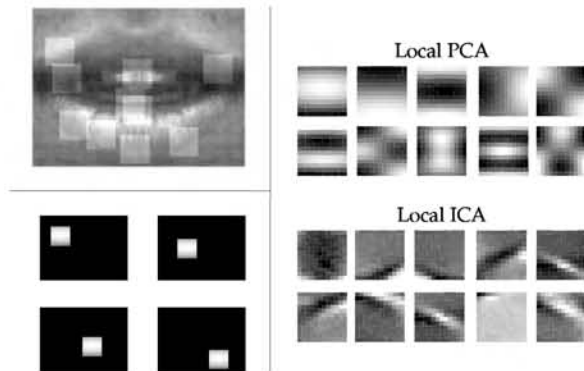

Figure 3: Upper left: Lip patches (12 pixels × 12 pixels) from randomly chosen locations used to develop local PCA and local ICA kernels. Lower left: Four orthogonal images generated from a single local PCA kernel. Right: Top 10 Local PCA and ICA kernels ordered according to projected variance (highest at top left). Note how the ICA vectors tend to be more local and consistent with the receptive fields found in V1.

the representation they may facilitate the job of the recognition engine by decreasing the statistical dependency amongst the coordinates. First each image in the database was projected onto the space spanned by the first 50 eigenvectors of the pixelwise covariance matrix. Then ICA was performed on the 50 PCA coordinate variables to obtain a new 50-dimensional non-orthogonal basis.

**ICA II:** A different approach to ICA was explored in [1] for face recognition tasks and by [6] for fMRI images. While in ICA-I the goal is to develop independent image coordinates, in ICA-II the goal is for the image bases themselves to be independent. Here independence of images is defined with respect to a probability space in which pixels are seen as outcomes and images as random vectors of such outcomes. The approach, which is described in detail in [6], resulted in a set of 50 images which were a non-orthogonal linear transformation of the first 50 eigenvectors of the pixelwise covariance matrix. The first 5 images (accounting for the largest amounts of projected variance) obtained via this approach to ICA are shown in the third row of Figure 2. The fourth row shows their magnitude spectrum. As reported in [1] the images obtained using this method are more local than those obtained via PCA.

## 4 Local Methods

Padgett et al. [8] reported surprisingly good results on an emotion recognition tasks using PCA on random patches of the face instead of the entire face. Recent theoretical work also places emphasis on spatially localized, wavelet-like image bases. One potential advantage of spatially localized image bases is that they provide explicit information about where things are happening, not just about what is happening. This facilitates the work of recognition engines on some tasks but the theoretical reasons for this are unclear at this point.

Local PCA and ICA kernels were developed based on a database of 18680 small patches (12 pixel × 12 pixel) chosen from random locations in the Tulip1s database. A sample of these random patches (superimposed on a lip image) is shown in the top panel of Figure 3. Hereafter we refer to the 12 pixel × 12 pixel images obtained

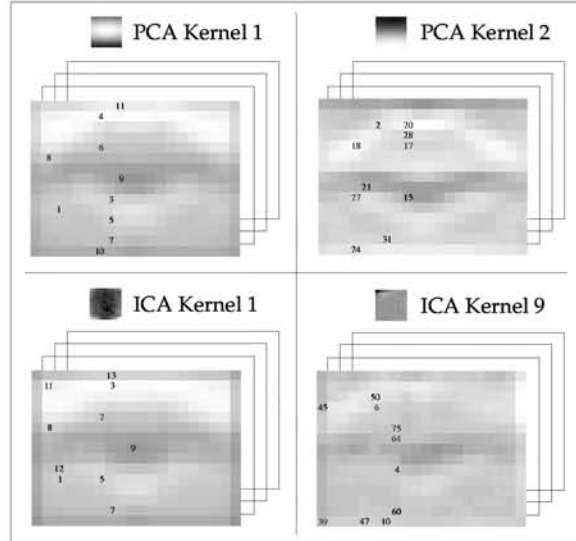

Figure 4: Kernel-location combinations chosen using unblocked variable selection. Top of each quadrant: Local ICA or PCA kernel. Bottom of each quadrant: Lip image convolved with corresponding local kernel, then downsampled. The numbers on the lip image indicate the order in which variables were chosen for the multiple regression procedure. There are no numbers on the right side of the lip images because only half of each lip image was used for the representation (since the images are symmetrized).

via PCA or ICA as "kernels". Image bases were generated by centering a local PCA or ICA kernel onto different locations and padding the rest of the matrix with zeros, as displayed in Figure 3 (lower left panel). This results on bases images which are local in space (the energy is localized about a single patch) and shifted versions of each other. The process of obtaining image coordinates can be seen as a filtering operation followed by subsampling: First the images are filtered using a bank of filters whose impulse response are the kernels obtained via PCA (or ICA). The relevant coordinates are obtained by subsampling at 300 uniformly distributed locations (15 locations vertically by 20 locations horizontally). We explored four different filtering approaches: (1) Single linear shift invariant filter (LSI); (2) Single linear shift variant filter (LSV); (3) Bank of LSI filters with blocked selection; (4) Bank of LSI filters combined with unblocked selection.

For the single-filter LSI approach, the images were convolved with a *single* local ICA kernel or a local PCA kernel. The top 5 local PCA and ICA kernels were each tested separately and the results obtained with the best of the 5 kernels were reported. For the single LSV-filtering approach different local PCA kernels were derived for a total of 117 non-overlapping regions each of which occupied 5 × 5 pixels. Each region of the 934 images was projected onto the first principal component corresponding to that location. This effectively resulted in an LSV filtering operation.

## 4.1 Automatic Selection of Focal Points

Padgett's [8] most successful method was based on outputs of local filters at manually selected focal regions. Their task was emotion recognition and the focal regions were the eyes and mouth. In visual speech recognition once the lips are chosen it

| | Image Processing | Performance ± s.e.m. |
|---|---|---|
| Global Methods | Global PCA | 79.2 ± 4.7 |
| | Global ICA I | 61.5 ± 4.5 |
| | Global ICA II | 74.0 ± 5.4 |
| Local Methods | Single-Filter LSI PCA | 90.6 ± 3.1 |
| | Single-Filter LSI ICA | 89.6 ± 3.0 |
| | Blocked Filter Bank PCA | 85.4 ± 3.7 |
| | Blocked Filter Bank ICA | 85.4 ± 3.0 |
| | Unblocked Filter Bank PCA | 91.7 ± 2.8 |
| | Unblocked Filter Bank ICA | 91.7 ± 3.2 |

Table 1: Best generalization performance (% correct) ± standard error of the mean for all image representations.

is unclear which regions would be most informative. Thus we developed a method for automatic selection of focal regions.

First 10 filters were developed via local ICA (or PCA). Each image was filtered using the 10-filter bank and the outputs were subsampled at 150 locations for a 1500 dimensional representation (10 filters × 150 locations) of each of the images in the dataset. Regions and variables of interest were then selected using a stepwise forward multiple regression procedure. First we choose the variable that, when averaging across the entire database, best reconstructed the original images. Here best reconstruction is defined in terms of least squares using a multiple regression model. Once a variable is selected, it is "tenured" and we search for the variable which in combination with the tenured ones best reconstructs the image database. The procedure is stopped when the number of tenured variables reaches a criterion point. We compared performance using 50, 100, and 150 tenured variables and report results with the best of those three numbers. We tested two different selection procedures, one blocked by location and one in which location was not blocked. In the first method the selection was done in blocks of 10 variables where each block contained the outputs of all the filters at a specific location. If a location was chosen, the outputs of the 10 filters in that location were automatically included in the final image representation. In the second method selection of variables was not blocked by location.

Figure 4 shows, for 2 local PCA and 2 local ICA kernels, the first 10 variables chosen for each particular kernel using the forward selection multiple regression procedure. The numbers on the lip images in this figure indicate the order in which particular kernel/location variables were chosen using the sequential regression procedure: "1" indicates the first variable chosen, "2" the second, etc.

## 5 Results and Conclusions

Table 1 shows the best generalization performance (out of the 9 HMM architectures tested) for each of the eight image representation methods. The local decompositions significantly outperformed the global ones ($t(106) = 4.10$, $p < 0.001$). The improved performance of local representations is consistent with current ideas on the importance of localized wavelet-like representations. However, it is unclear why local decompositions work better. One possibility is that these results apply only to this particular recognition engine and the problem at hand (i.e., hidden Markov models for speechreading). Yet similar results with local representations were reported in [8] on an emotion classification task with a 3 layer backpropaga-

tion network and in [1] on an expression classification tasks with a nearest neighbor classifier. Another possible explanation for the advantage of local representations is that global unsupervised decompositions emphasize subject identity while local decompositions tend to hide it. We found some evidence consistent with this idea by testing global and local representations on a subject identification task (i.e., recognizing which person the lip images belong to). For this task the global representations outperformed the local ones. However this result is inconsistent with [8] which found local representations were better on emotion classification and on subject identification tasks. Another possibility is that local representations make more explicit information about where things are happening, not just what is happening, and such information turns out to be important for the task at hand.

The image representations obtained using the bank of filter methods with unblocked selection yielded the best results. The stepwise regression technique used to select kernels and regions of interest led to substantial gains in recognition performance. In fact the highest generalization performance reported here (91.7% with the bank of filters using unblocked variable selection) surpassed the best published performance on this dataset [5].

# References

[1] M.S. Bartlett. *Face Image Analysis by Unsupervised Learning and Redundancy Reduction*. PhD thesis, University of California, San Diego, 1998.

[2] M.S. Bartlett, P.A. Viola, T.J. Sejnowski, J. Larsen, J. Hager, and P. Ekman. Classifying facial action. In D. Touretski, M. Mozer, and M. Hasselmo, editors, *Advances in Neural Information Processing Systems*, volume 8, pages 823–829. Morgan Kaufmann, San Mateo, CA, 1996.

[3] A.J. Bell and T.J. Sejnowski. An information-maximization approach to blind separation and blind deconvolution. *Neural Computation*, 7(6):1129–1159, 1995.

[4] G. Cottrell and J. 1991 Metcalfe. Face, gender and emotion recognition using holons. In D. Touretzky, editor, *Advances in Neural Information Processing Systems*, volume 3, pages 564–571, San Mateo, CA, 1991. Morgan Kaufmann.

[5] Juergen Luettin. *Visual Speech and Speaker Recognition*. PhD thesis, University of Sheffield, 1997.

[6] M.J. McKeown, S. Makeig, G.G. Brown, T-P. Jung, S.S. Kindermann, A.J. Bell, and T.J. Sejnowski. Analysis of fmri data by decomposition into independent components. *Proc. Nat. Acad. Sci.*, in press.

[7] J.R. Movellan. Visual speech recognition with stochastic networks. In G. Tesauro, D.S. Touretzky, and T. Leen, editors, *Advances in Neural Information Processing Systems*, volume 7, pages 851–858. MIT Press, Cambridge, MA, 1995.

[8] C. Padgett and G. Cottrell. Representing face images for emotion classification. In M. Mozer, M. Jordan, and T. Petsche, editors, *Advances in Neural Information Processing Systems*, volume 9, Cambridge, MA, 1997. MIT Press.

[9] M. Turk and A. Pentland. Eigenfaces for recognition. *Journal of Cognitive Neuroscience*, 3(1):71–86, 1991.
